# Associative memory in realistic neuronal networks

**P.E. Latham**[*]
Department of Neurobiology
University of California at Los Angeles
Los Angeles, CA 90095
*pel@ucla.edu*

## Abstract

Almost two decades ago, Hopfield [1] showed that networks of highly reduced model neurons can exhibit multiple attracting fixed points, thus providing a substrate for associative memory. It is still not clear, however, whether realistic neuronal networks can support multiple attractors. The main difficulty is that neuronal networks *in vivo* exhibit a stable background state at low firing rate, typically a few Hz. Embedding attractor is easy; doing so without destabilizing the background is not. Previous work [2,3] focused on the sparse coding limit, in which a vanishingly small number of neurons are involved in any memory. Here we investigate the case in which the number of neurons involved in a memory scales with the number of neurons in the network. In contrast to the sparse coding limit, we find that multiple attractors can co-exist robustly with a stable background state. Mean field theory is used to understand how the behavior of the network scales with its parameters, and simulations with analog neurons are presented.

One of the most important features of the nervous system is its ability to perform associative memory. It is generally believed that associative memory is implemented using attractor networks – experimental studies point in that direction [4–7], and there are virtually no competing theoretical models. Perhaps surprisingly, however, it is still an open theoretical question whether attractors can exist in realistic neuronal networks. The "realistic" feature that is probably hardest to capture is the steady firing at low rates – the background state – that is observed throughout the intact nervous system [8–13]. The reason it is difficult to build an attractor network that is stable at low firing rates, at least in the sparse coding limit, is as follows [2,3]:

Attractor networks are constructed by strengthening recurrent connections among sub-populations of neurons. The strengthening must be large enough that neurons within a sub-population can sustain a high firing rate state, but not so large that the sub-population can be spontaneously active. This implies that the neuronal gain functions – the firing rate of the post-synaptic neurons as a function of the average

---

[*]http://culture.neurobio.ucla.edu/~pel

firing rate of the pre-synaptic neurons – must be sigmoidal: small at low firing rate to provide stability, high at intermediate firing rate to provide a threshold (at an unstable equilibrium), and low again at high firing rate to provide saturation and a stable attractor. In other words, a requirement for the co-existence of a stable background state and multiple attractors is that the gain function of the excitatory neurons be superlinear at the observed background rates of a few Hz [2, 3]. However – and this is where the problem lies – above a few Hz most realistic gain function are nearly linear or sublinear (see, for example, Fig. B1 of [14]).

The superlinearity requirement rests on the implicit assumption that the activity of the sub-population involved in a memory does not affect the other neurons in the network. While this assumption is valid in the sparse coding limit, it breaks down in realistic networks containing both excitatory and inhibitory neurons. In such networks, activity among excitatory cells results in inhibitory feedback. This feedback, if powerful enough, can stabilize attractors even without a saturating nonlinearity, essentially by stabilizing the equilibrium (above considered unstable) on the steep part of the gain function. The price one pays, though, is that a reasonable fraction of the neurons must be involved in each of the memories, which takes us away from the sparse coding limit and thus reduces network capacity [15].

# 1 The model

A relatively good description of neuronal networks is provided by synaptically coupled, conductance-based neurons. However, because communication is via action potentials, such networks are difficult to analyze. An alternative is to model neurons by their firing rates. While this is unlikely to capture the full temporal network dynamics [16], it is useful for studying equilibria. In such simplified models, the equilibrium firing rate of a neuron is a function of the firing rates of all the other neurons in the network. Letting $\nu_{Ei}$ and $\nu_{Ii}$ denote the firing rates of the excitatory and inhibitory neurons, respectively, and assuming that synaptic input sums linearly, the equilibrium equations may be written

$$\nu_{Ei} = \phi_{Ei}\left(\sum_j A_{ij}^{EE}\nu_{Ej}, \sum_j A_{ij}^{EI}\nu_{Ij}\right) \qquad (1a)$$

$$\nu_{Ii} = \phi_{Ii}\left(\sum_j A_{ij}^{IE}\nu_{Ej}, \sum_j A_{ij}^{II}\nu_{Ij}\right). \qquad (1b)$$

Here $\phi_E$ and $\phi_I$ are the excitatory and inhibitory gain functions and $A_{ij}$ determines the connection strength from neuron $j$ to neuron $i$. The gain functions can, in principle, be derived from conductance-based model equations [17].

Our goal here is to determine under what conditions Eq. (1) allows both attractors and a stable state at low firing rate. To accomplish this we will use mean field theory. While this theory could be applied to the full set of equations, to reduce complexity we make a number of simplifications. First, we let the inhibitory neurons be completely homogeneous ($\phi_{Ii}$ independent of $i$ and connectivity to and from inhibitory neurons all-to-all and uniform). In that case, Eq. (1b) becomes simply $\nu_I = \phi(\nu_E, \nu_I)$ where $\nu_E$ and $\nu_I$ are the average firing rates of the excitatory and inhibitory neurons. Solving for $\nu_I$ and inserting the resulting expression into Eq. (1a) results in the expression $\nu_{Ei} = \phi_{Ei}\left(\sum_j A_{ij}^{EE}\nu_{Ej}, A^{EI}\nu_I(\nu_E)\right)$ where $A^{EI} \equiv \sum_j A_{ij}^{EI}$.

Second, we let $\phi_{Ei}$ have the form $\phi_{Ei}(u,v) = \phi_E(x_i + bu - cv)$ where $x_i$ is a Gaussian random variable, and similarly for $\phi_I$ (except with different constants $b$ and $c$ and no dependence on $i$). Finally, we assume that $\phi_I$ is threshold linear and the network operates in a regime in which the inhibitory firing rate is above zero. With these simplifications, and a trivial redefinition of constants, Eq. (1a) becomes

$$\nu_i = \beta p^{1/2} \phi \left( x_i - (a+1)\nu + \sum_j A_{ij}\nu_j \right).$$
(2)

We have dropped the sub and superscript $E$, since Eq. (2) refers exclusively to excitatory neurons, defined $\nu$ to be the average firing rate, $\nu \equiv N^{-1}\sum_i \nu_i$, and rescaled parameters. We let the function $\phi$ be $\mathcal{O}(1)$, so $\beta$ can be interpreted as the gain. The parameter $p$ is the number of memories. The reduction from Eq. (1) to Eq. (2) was done solely to simplify the analysis; the techniques we will use apply equally well to the general case, Eq. (1).

Note that the gain function in Eq. (2) decreases with increasing average firing rate, since it's argument is $-(1+a)\nu$ and $a$ is positive. This negative dependence on $\nu$ arises because we are working in the large coupling regime in which excitation and inhibition are balanced [18, 19]. The negative coupling to firing rate has important consequences for stability, as we will see below.

We let the connectivity matrix have the form

$$A_{ij} = \frac{1}{\langle g \rangle N} C_{ij} g(W_{ij} + J_{ij}).$$

Here $N$ is the number of excitatory neurons; $C_{ij}$, which regulates the degree of connectivity, is $1/c$ with probability $c$ and and 0 with probability $(1-c)$ (except $C_{ii} = 0$, meaning no autapses); $g(z)$ is an $\mathcal{O}(1)$ clipping function that keeps weights from falling below zero or getting too large; $\langle g \rangle$ is the mean value of $g(z)$, defined in Eq. (4) below; $W_{ij}$, which corresponds to background connectivity, is a random matrix whose elements are Gaussian distributed with mean 1 and variance $\delta w^2$; and $J_{ij}$ produces the attractors. We will follow the Hopfield prescription and write $J_{ij}$ as

$$J_{ij} = \frac{\epsilon}{\sqrt{p}} \sum_{\mu=1}^{p} \eta_{\mu i}\eta_{\mu j}$$
(3)

where $\epsilon$ is the coupling strength among neurons involved in the memories, and the patterns $\eta_{\mu i}$ determine which neurons participate in each memory. The $\eta_{\mu i}$ are a set of uncorrelated vectors with zero mean and unit variance. In simulations we use $\eta_{\mu i} = [(1-f)/f]^{1/2}$ with probability $f$ and $-[f/(1-f)]^{1/2}$ with probability $1-f$, so a fraction $f$ of the neurons are involved in each memory. Other choices are unlikely to significantly change our results.

## 2 Mean field equations

The main difficulty in deriving the mean field equations from Eq. (2) is separating the signal from the noise. Our first step in this endeavor is to analyze the noise

associated with the clipped weights. To do this we break $C_{ij}g(W_{ij} + J_{ij})$ into two pieces: $C_{ij}g(W_{ij} + J_{ij}) = \langle g \rangle + \langle g' \rangle J_{ij} + \delta C_{ij}$ where

$$\delta C_{ij} \equiv C_{ij}g(W_{ij} + J_{ij}) - (\langle g \rangle + \langle g' \rangle J_{ij}).$$

The angle brackets around $g$ represent an average over the distributions of $W_{ij}$ and $J_{ij}$, and a prime denotes a derivative. In the large $p$ limit, $\delta C_{ij}$ can be treated as a random matrix whose main role is to increase the effective noise [20]. The mean of $\delta C_{ij}$ is zero and its variance normalized to $\langle g \rangle^2 / c$, which we denote $\sigma^2$, is given by

$$\sigma^2 \equiv \frac{c}{\langle g \rangle^2} \text{Var}[\delta C_{ij}] = \frac{\langle g^2 \rangle - c(1 + \langle g' \rangle^2 \langle J_{ij}^2 \rangle)}{\langle g \rangle^2}.$$

For large $p$, the elements of $J_{ij}$ are Gaussian with zero mean and variance $\epsilon^2$, so the averages involving $g$ can be written

$$\langle g^k \rangle = \int dz \, \frac{\exp[-z^2/2(\delta w^2 + \epsilon^2)]}{[2\pi(\delta w^2 + \epsilon^2)]^{1/2}} g^k(1 + z) \qquad (4)$$

where $k$ can be either an exponent or a prime and the "1" in $g(1 + z)$ corresponds to the mean of $W_{ij}$. In our simulations we use the clipping function $g(z) = z$ if $z$ is between 0 and 2, 0 if $z \leq 0$ and 2 if $z \geq 2$.

Our main assumptions in the development of a mean field theory are that $\sum_{j \neq i} \delta C_{ij} \nu_j$ is a Gaussian random variable, and that $\delta C_{ij}$ and $\nu_j$ are independent. Consequently,

$$\text{Var}\left[ \frac{1}{\langle g \rangle N} \sum_{j \neq i} \delta C_{ij} \nu_j \right] = \frac{\sigma^2}{cN} \langle \nu^2 \rangle$$

where $\langle \nu^2 \rangle \equiv N^{-1} \sum_i \nu_i^2$ is the second moment of the firing rate. Letting $\hat{\theta}_i$ be a zero mean Gaussian random variable with variance $\hat{\theta}^2 \equiv \sigma^2 \langle \nu^2 \rangle / cN$, we can use the above assumptions along with the definition of $J_{ij}$, Eq. (3), to write Eq. (20) as

$$\nu_i = \beta p^{1/2} \phi \left( x_i - a\nu + p^{-1/2} \epsilon_c \eta_{\mu i} \frac{1}{N} \sum_{j \neq i} \eta_{\mu j} \nu_j + \hat{\theta}_i \right). \qquad (5)$$

We have defined the clipped memory strength, $\epsilon_c$, as $\epsilon_c \equiv \epsilon \langle g' \rangle / \langle g \rangle$. While it is not totally obvious from the above equations, it can be shown that both $\sigma^2$ and $\epsilon_c$ become independent of $\epsilon$ for large $\epsilon$. This makes network behavior robust to changes in $\epsilon$, the strength of the memories, so long as $\epsilon$ is large.

Derivation of the mean field equations from Eq. (5) follow standard methods [21, 22]. For definiteness we take $\phi(x)$ to be threshold linear: $\phi(x) = \max(0, x)$. For the case of one active memory, the mean field equations may then be written in the form

$$w = \frac{\beta \epsilon_c}{1-r} \Delta F_1(w,z) \tag{6a}$$

$$1 = \alpha \frac{\beta^2 \epsilon_c^2}{(1-r)^2} \left[ \frac{\sigma^2}{c\epsilon_c^2} + \frac{1}{(1-q)^2} \right] [F_2(z) + f\Delta F_2(w,z)] \tag{6b}$$

$$+ \frac{\beta^2 \theta_0^2 a^2 / x_0^2}{(1-r)^2} [F_1(z) + f\Delta F_1(w,z)]^2$$

$$r = \frac{\alpha \beta \epsilon_c q}{1-q} \tag{6c}$$

$$q = \frac{\beta \epsilon_c}{1 + \alpha \beta \epsilon_c} [F_0(z) + f\Delta F_0(w,z)] \tag{6d}$$

where $\alpha \equiv p/N$ is the load parameter, $x_0$ and $\theta_0^2/p$ are the mean and variance of of $x_i$ (see Eq. (2)), and, recall, $f$ is the fraction of neurons that participate in each memory. The functions $F_k$ and $\Delta F_k$ are defined by

$$F_k(z) \equiv \int_{-z}^{\infty} \frac{d\xi}{(2\pi)^{1/2}} (z+\xi)^k \exp(-\xi^2/2)$$

$$\Delta F_k(w,z) \equiv F_k(w+z) - F_k(z).$$

For large negative $z$, $F_k(z)$ vanishes as $\exp(-z^2/2)$, while for large positive $z$, $F_k(z) \to z^k/k!$.

The average firing rate, $\nu$, and strength of the memory, $m \equiv N^{-1} \sum_i \eta_{1j} \nu_j$ (taken without loss of generality to be the overlap with pattern 1), are given in terms of $z$ and $w$ as

$$\nu = \frac{x_0}{a + p^{-1/2} \epsilon_c (z/w + f) \Delta F_1(w,z) / (F_1(z) + f\Delta F_1(w,z))}$$

$$m = \frac{(1-f)\Delta F_1(w,z)}{F_1(z) + f\Delta F_1(w,z)} \nu.$$

## 3 Results

The mean field equations can be understood by examining Eqs. (6a) and (6b). The first of these, Eq. (6a), is a rescaled form of the equation for the overlap, $m$. (From the definition of $\Delta F_1$ given above, it can be seen that $m$ is proportional to $w$ for small $w$). This equation always has a solution at $w = 0$ (and thus $m = 0$), which corresponds to a background state with no memories active. If $\beta \epsilon_c$ is large enough, there is a second solution with $w$ (and thus $m$) greater than zero. This second solution corresponds to a memory. The other relevant equation, Eq. (6b), describes the behavior of the mean firing rate. This equation looks complicated only because the noise – the variation in firing rate from neuron to neuron – must be determined self-consistently.

The solutions to Eqs. (6a) and (6b) are plotted in Fig. 1 in the $z$-$w$ plane. The solid lines, including the horizontal line at $w = 0$, represents the solution to Eq. (6a), the

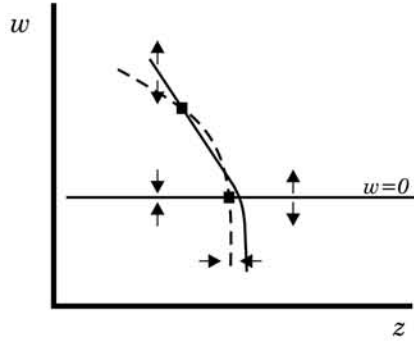

Figure 1: Graphical solution of Eqs. (6a) and (6b). Solid lines, including the one at $w = 0$: solution to Eq. (6a). Dashed line: solution to Eq. (6b). The arrows indicate approximate flow directions: vertical arrows indicate time evolution of $w$ at fixed $z$; horizontal arrows indicate time evolution of $z$ at fixed $w$. The black squares show potentially stable fixed points. Note the exchange of stability to the right of the solid curve, indicating that intersections too far to the right will be unstable.

dashed line the solution to Eq. (6b), and their intersections solutions to both. While stability cannot be inferred from the equilibrium equations, a reasonable assumption is that the evolution equations for the firing rates, at least near an equilibrium, have the form $\tau d\nu_i/dt = \phi_i - \nu_i$. In that case, the arrows represent flow directions, and we see that there are potentially stable equilibria at the intersections marked by the solid squares.

Note that in the sparse coding limit, $f \to 0$, $z$ is independent of $w$, meaning that the mean firing rate, $\nu$, is independent of the overlap, $m$. In this limit there can be no feedback to inhibitory neurons, and thus no chance for stabilization. In terms of Fig. 1, the effect of letting $f \to 0$ is to make the dashed line vertical. This eliminates the possibility of the upper stable equilibrium (the solid square at $w > 0$), and returns us to the situation where a superlinear gain function is required for attractors to be embedded, as discussed in the introduction.

Two important conclusions can be drawn from Fig. 1. First, the attractors can be stable even though the gain functions never saturate (recall that we used threshold-linear gain functions). The stabilization mechanism is feedback to inhibitory neurons, via the $-(1 + a)\nu$ term in Eq. (2). This feedback is what makes the dashed line in Fig. 1 bend, allowing a stable equilibrium at $w > 0$. Second, if the dashed line shifts to the right relative to the solid line, the background becomes destabilized. This is because there is an exchange of stability, as indicated by the arrows. Thus, there is a tradeoff: $w$, and thus the mean firing rate of the memory neurons, can be increased by shifting the dashed line up or to the right, but eventually the background becomes destabilized. Shifting the dashed line to the left, on the other hand, will eventually eliminate the solution at $w > 0$, destroying all attractors but the background.

For fixed load parameter $\alpha$, fraction of neurons involved in a memory, $f$, and degree of connectivity, $c$, there are three parameters that have a large effect on the location of the equilibria in Fig. 1: the gain, $\beta$, the clipped memory strength, $\epsilon_c$, and the degree of heterogeneity in individual neurons, $\theta_0$. The effect of the first two can be seen in Fig. 2, which shows a stability plot in the $\epsilon$-$\beta$ plane, determined by numerically solving the the equations $\tau d\nu_i/dt = \phi_i - \nu_i$ (see Eq. (2)). The filled circles indicate regions where memories were embedded without destabilizing the background, open circles indicate regions where no memories could be embedded, and ×s indicate regions where the background was unstable. As discussed above, $\epsilon_c$ becomes approximately independent of the strength of the memories, $\epsilon$, when $\epsilon$ becomes large. This is seen in Fig. 2A, in which network behavior stabilizes when $\epsilon$ becomes larger than about 4; increasing $\epsilon$ beyond 8 would, presumably,

produce no surprises. There is some sensitivity to gain, $\beta$: when $\epsilon > 4$, memories co-existed with a stable background for $\beta$ in a $\pm15\%$ range. Although not shown, the same was true of $\theta_0$: increasing it by about 20% eliminated the attractors; decreasing it by the same amount destabilized the background. However, more detailed analysis indicates that the stability region gets larger as the number of neurons in the network, $N$, increases. This is because fluctuations destabilize the background, and those fluctuations fall off as $N^{-1/2}$.

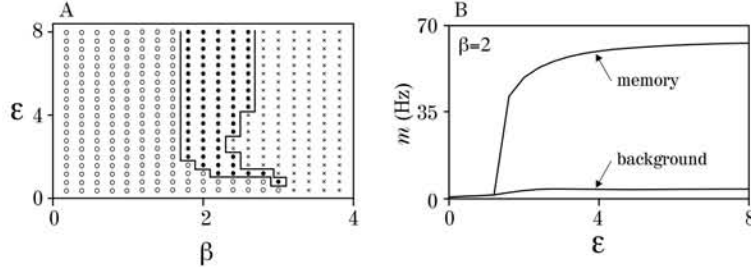

Figure 2: **A**. Stability diagram, found by solving the set of equations $\tau d\nu_i/dt = \phi_i - \nu_i$ with the argument of $\phi_i$ given in Eq. (2). Filled circles: memories co-exist with a stable background (also outlined with solid lines); open circles: memories could not be embedded; ×s: background was unstable. The average background rate, when the background was stable, was around 3 Hz. The network parameters were $\theta_0 = 6$, $x_0 = 1.5$, $a = 0.5$, $c = 0.3$, $\alpha = 2.5\%$, and $\delta w = 0.3$. 2000 neurons were used in the simulations. These parameters led to an effective gain, $p^{1/2}\beta\epsilon_c$, of about 10, which is consistent with the gain in large networks in which each neuron receives $\sim$5-10,000 inputs. **B**. Plot of firing rate of memory neurons, $m$, when the memory was active (upper trace) and not active (lower trace) versus $\epsilon$ at $\beta = 2$.

## 4   Discussion

The main outcome of this analysis is that attractors can co-exist with a stable background when neurons have generic threshold-linear gain functions, so long as the sparse coding limit is avoided. The parameter regime for this co-existence is much larger than for attractor networks that operate in the sparse coding limit [2, 23]. While these results are encouraging, they do not definitively establishing that attractors can exist in realistic networks. Future work must include inhibitory neurons, incorporate a much larger exploration of parameter space to ensure that the results are robust, and ultimately involve simulations with spiking neurons.

## 5   Acknowledgements

This work was supported by NIMH grant #R01 MH62447.

## References

[1] J.J. Hopfield. Neural networks and physical systems with emergent collective computational abilities. *Proc. Natl. Acad. Sci.*, 79:2554–2558, 1982.

[2] N. Brunel. Persistent activity and the single-cell frequency-current curve in a cortical network model. *Network: Computation in Neural Systems*, 11:261–280, 2000.

[3] P.E. Latham and S.N. Nirenberg. Intrinsic dynamics in cultured neuronal networks. *Soc. Neuroscience Abstract*, 25:2259, 1999.

[4] J.M. Fuster and G.E. Alexander. Neuron activity related to short-term memory. *Science*, 173:652–654, 1971.

[5] Y. Miyashita. Inferior temporal cortex: where visual perception meets memory. *Annu Rev Neurosci*, 16:245–263, 1993.

[6] P.S. Goldman-Rakic. Cellular basis of working memory. *Neuron*, 14:477–485, 1995.

[7] R. Romo, C.D. Brody, A. Hernandez, and L. Lemus. Neuronal correlates of parametric working memory in the prefrontal cortex. *Nature*, 399:470–473, 1999.

[8] C.D. Gilbert. Laminar differences in receptive field properties of cells in cat primary visual cortex. *J. Physiol.*, 268:391–421, 1977.

[9] Y. Lamour, P. Dutar, and A. Jobert. Cerebral neocortical neurons in the aged rat: spontaneous activity, properties of pyramidal tract neurons and effect of acetylcholine and cholinergic drugs. *Neuroscience*, 16:835–844, 1985.

[10] M.B. Szente, A. Baranyi, and C.D. Woody. Intracellular injection of apamin reduces a slow potassium current mediating afterhyperpolarizations and IPSPs in neocortical neurons of cats. *Brain Res.*, 461:64–74, 1988.

[11] I. Salimi, H.H. Webster, and R.W. Dykes. Neuronal activity in normal and deafferented forelimb somatosensory cortex of the awake cat. *Brain Res.*, 656:263–273, 1994.

[12] J.F. Herrero and P.M. Headley. Cutaneous responsiveness of lumbar spinal neurons in awake and halothane-anesthetized sheep. *J. Neurophysiol.*, 74:1549–1562, 1997.

[13] K. Ochi and J.J. Eggermont. Effects of quinine on neural activity in cat primary auditory cortex. *Hear. Res.*, 105:105–18, 1997.

[14] P.E. Latham, B.J. Richmond, P.G. Nelson, and S.N. Nirenberg. Intrinsic dynamics in neuronal networks. I. Theory. *J. Neurophysiol.*, 83:808–827, 2000.

[15] M.V. Tsodyks and M.V. Feigel'man. The enhanced storage capacity in neural networks with low activity level. *Europhys. Lett.*, 6:101–105, 1988.

[16] A. Treves. Mean-field analysis of neuronal spike dynamics. *Network*, 4:259–284, 1993.

[17] O. Shriki, D. Hansel, and H. Sompolonski. Modeling neuronal networks in cortex by rate models using the current-frequency response properties of cortical cells. *Soc. Neuroscience Abstract*, 24:143, 1998.

[18] C. van Vreeswijk and H. Sompolinsky. Chaos in neuronal networks with balanced excitatory and inhibitory activity. *Science*, 274:1724–1726, 1996.

[19] C. van Vreeswijk and H. Sompolinsky. Chaotic balanced state in a model of cortical circuits. *Neural Comput.*, 10:1321–1371, 1998.

[20] H. Sompolinsky. Neural networks with nonlinear synapses and a static noise. *Phys. Rev. A*, 34:2571–2574, 1986.

[21] J. Hertz, A. Krogh, and R.G. Palmer. *Introduction to the theory of neural computation*. Addison Wesley, Redwood City, CA, 1991.

[22] A.N. Burkitt. Retrieval properties of attractor neural that obey Dale's law using a self-consistent signal-to-noise analysis. *Network: Computation in Neural Systems*, 7:517–531, 1996.

[23] D.J. Amit and N. Brunel. Dynamics of a recurrent network of spiking neurons before and following learning. *Network*, 8:373–404, 1997.
